# Off-policy Learning with Options and Recognizers

**Doina Precup**
McGill University
Montreal, QC, Canada

**Richard S. Sutton**
University of Alberta
Edmonton, AB, Canada

**Cosmin Paduraru**
University of Alberta
Edmonton, AB, Canada

**Anna Koop**
University of Alberta
Edmonton, AB, Canada

**Satinder Singh**
University of Michigan
Ann Arbor, MI, USA

## Abstract

We introduce a new algorithm for off-policy temporal-difference learning with function approximation that has lower variance and requires less knowledge of the behavior policy than prior methods. We develop the notion of a *recognizer*, a filter on actions that distorts the behavior policy to produce a related target policy with low-variance importance-sampling corrections. We also consider target policies that are deviations from the state distribution of the behavior policy, such as potential temporally abstract options, which further reduces variance. This paper introduces recognizers and their potential advantages, then develops a full algorithm for linear function approximation and proves that its updates are in the same direction as on-policy TD updates, which implies asymptotic convergence. Even though our algorithm is based on importance sampling, we prove that it requires absolutely no knowledge of the behavior policy for the case of state-aggregation function approximators.

Off-policy learning is learning about one way of behaving while actually behaving in another way. For example, Q-learning is an off- policy learning method because it learns about the optimal policy while taking actions in a more exploratory fashion, e.g., according to an $\varepsilon$-greedy policy. Off-policy learning is of interest because only one way of selecting actions can be used at any time, but we would like to learn about many different ways of behaving from the single resultant stream of experience. For example, the options framework for temporal abstraction involves considering a variety of different ways of selecting actions. For each such option one would like to learn a model of its possible outcomes suitable for planning and other uses. Such option models have been proposed as fundamental building blocks of grounded world knowledge (Sutton, Precup & Singh, 1999; Sutton, Rafols & Koop, 2005). Using off-policy learning, one would be able to learn predictive models for many options at the same time from a single stream of experience.

Unfortunately, off-policy learning using temporal-difference methods has proven problematic when used in conjunction with function approximation. Function approximation is essential in order to handle the large state spaces that are inherent in many problem do-

mains. Q-learning, for example, has been proven to converge to an optimal policy in the tabular case, but is unsound and may diverge in the case of linear function approximation (Baird, 1996). Precup, Sutton, and Dasgupta (2001) introduced and proved convergence for the first off-policy learning algorithm with linear function approximation. They addressed the problem of learning the expected value of a target policy based on experience generated using a different behavior policy. They used importance sampling techniques to reduce the off-policy case to the on-policy case, where existing convergence theorems apply (Tsitsiklis & Van Roy, 1997; Tadic, 2001). There are two important difficulties with that approach. First, the behavior policy needs to be stationary and known, because it is needed to compute the importance sampling corrections. Second, the importance sampling weights are often ill-conditioned. In the worst case, the variance could be infinite and convergence would not occur. The conditions required to prevent this were somewhat awkward and, even when they applied and asymptotic convergence was assured, the variance could still be high and convergence could be slow.

In this paper we address both of these problems in the context of off-policy learning for options. We introduce the notion of a *recognizer*. Rather than specifying an explicit target policy (for instance, the policy of an option), about which we want to make predictions, a recognizer specifies a condition on the actions that are selected. For example, a recognizer for the temporally extended action of picking up a cup would not specify which hand is to be used, or what the motion should be at all different positions of the cup. The recognizer would recognize a whole variety of directions of motion and poses as part of picking the cup. The advantage of this strategy is not that one might prefer a multitude of different behaviors, but that the behavior may be based on a variety of different strategies, all of which are relevant, and we would like to learn from any of them. In general, a recognizer is a function that recognizes or accepts a space of different ways of behaving and thus, can learn from a wider range of data.

Recognizers have two advantages over direct specification of a target policy: 1) they are a natural and easy way to specify a target policy for which importance sampling will be well conditioned, and 2) they do not require the behavior policy to be known. The latter is important because in many cases we may have little knowledge of the behavior policy, or a stationary behavior policy may not even exist. We show that for the case of state aggregation, even if the behavior policy is unknown, convergence to a good model is achieved.

## 1   Non-sequential example

The benefits of using recognizers in off-policy learning can be most easily seen in a non-sequential context with a single continuous action. Suppose you are given a sequence of sample actions $a_i \in [0,1]$, selected i.i.d. according to probability density $b : [0,1] \mapsto \Re^+$ (the behavior density). For example, suppose the behavior density is of the oscillatory form shown as a red line in Figure 1. For each each action, $a_i$, we observe a corresponding outcome, $z_i \in \Re$, a random variable whose distribution depends only on $a_i$. Thus the behavior density induces an outcome density. The on-policy problem is to estimate the mean $m^b$ of the outcome density. This problem can be solved simply by averaging the sample outcomes: $\hat{m}^b = (1/n)\sum_{i=1}^n z_i$. The off-policy problem is to use this same data to learn what the mean would be if actions were selected in some way other than $b$, for example, if the actions were restricted to a designated range, such as between 0.7 and 0.9.

There are two natural ways to pose this off-policy problem. The most straightforward way is to be equally interested in all actions within the designated region. One professes to be interested in actions selected according to a target density $\pi : [0,1] \mapsto \Re^+$, which in the example would be 5.0 between 0.7 and 0.9, and zero elsewhere, as in the dashed line in

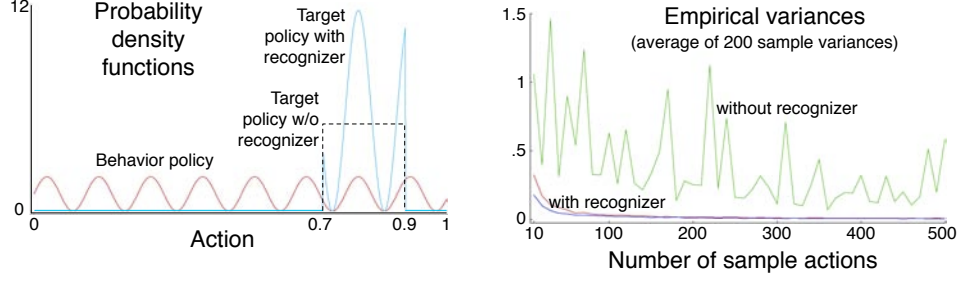

Figure 1: The left panel shows the behavior policy and the target policies for the formulations of the problem with and without recognizers. The right panel shows empirical estimates of the variances for the two formulations as a function of the number sample actions. The lowest line is for the formulation using empirically-estimated recognition probabilities.

Figure 1 (left). The importance- sampling estimate of the mean outcome is

$$\hat{m}^{\pi} = \frac{1}{n} \sum_{i=1}^{n} \frac{\pi(a_i)}{b(a_i)} z_i. \tag{1}$$

This approach is problematic if there are parts of the region of interest where the behavior density is zero or very nearly so, such as near 0.72 and 0.85 in the example. Here the importance sampling ratios are exceedingly large and the estimate is poorly conditioned (large variance). The upper curve in Figure 1 (right) shows the empirical variance of this estimate as a function of the number of samples. The spikes and uncertain decline of the empirical variance indicate that the distribution is very skewed and that the estimates are very poorly conditioned.

The second way to pose the problem uses recognizers. One professes to be interested in actions to the extent that they are both selected by $b$ and within the designated region. This leads to the target policy shown in blue in the left panel of Figure 1 (it is taller because it still must sum to 1). For this problem, the variance of (1) is much smaller, as shown in the lower two lines of Figure 1 (right). To make this way of posing the problem clear, we introduce the notion of a recognizer function $c : \mathcal{A} \mapsto \mathfrak{R}^{+}$. The action space in the example is $\mathcal{A} = [0, 1]$ and the recognizer is $c(a) = 1$ for $a$ between 0.7 and 0.9 and is zero elsewhere. The target policy is defined in general by

$$\pi(a) = \frac{c(a)b(a)}{\sum_x c(x)b(x)} = \frac{c(a)b(a)}{\mu}. \tag{2}$$

where $\mu = \sum_x c(x)b(x)$ is a constant, equal to the probability of recognizing an action from the behavior policy. Given $\pi$, $\hat{m}^{\pi}$ from (1) can be rewritten in terms of the recognizer as

$$\hat{m}^{\pi} = \frac{1}{n} \sum_{i=1}^{n} z_i \frac{\pi(a_i)}{b(a_i)} = \frac{1}{n} \sum_{i=1}^{n} z_i \frac{c(a_i)b(a_i)}{\mu} \frac{1}{b(a_i)} = \frac{1}{n} \sum_{i=1}^{n} z_i \frac{c(a_i)}{\mu} \tag{3}$$

Note that the target density does not appear at all in the last expression and that the behavior distribution appears only in $\mu$, which is independent of the sample action. If this constant is known, then this estimator can be computed with no knowledge of $\pi$ or $b$. The constant $\mu$ can easily be estimated as the fraction of recognized actions in the sample. The lowest line in Figure 1 (right) shows the variance of the estimator using this fraction in place of the recognition probability. Its variance is low, no worse than that of the exact algorithm, and apparently slightly lower. Because this algorithm does not use the behavior density, it can be applied when the behavior density is unknown or does not even exist. For example, suppose actions were selected in some deterministic, systematic way that in the long run produced an empirical distribution like $b$. This would be problematic for the other algorithms but would require no modification of the recognition-fraction algorithm.

## 2 Recognizers improve conditioning of off-policy learning

The main use of recognizers is in formulating a target density $\pi$ about which we can successfully learn predictions, based on the current behavior being followed. Here we formalize this intuition.

**Theorem 1** *Let $A = \{a_1, \ldots a_k\} \subseteq \mathcal{A}$ be a subset of all the possible actions. Consider a fixed behavior policy $b$ and let $\pi_A$ be the class of policies that only choose actions from $A$, i.e., if $\pi(a) > 0$ then $a \in A$. Then the policy induced by $b$ and the binary recognizer $c_A$ is the policy with minimum-variance one-step importance sampling corrections, among those in $\pi_A$:*

$$\pi \text{ as given by (2)} = \arg\min_{\pi \in \pi_A} E_b \left[ \left( \frac{\pi(a_i)}{b(a_i)} \right)^2 \right] \tag{4}$$

**Proof:** Denote $\pi(a_i) = \pi_i$, $b(a_i) = b_i$. Then the expected variance of the one-step importance sampling corrections is:

$$E_b \left[ \left( \frac{\pi_i}{b_i} \right)^2 \right] - E_b^2 \left[ \left( \frac{\pi_i}{b_i} \right) \right] = \sum_i b_i \left( \frac{\pi_i}{b_i} \right)^2 - 1 = \sum_i \frac{\pi_i^2}{b_i} - 1,$$

where the summation (here and everywhere below) is such that the action $a_i \in A$. We want to find $\pi_i$ that minimizes this expression, subject to the constraint that $\sum_i \pi_i = 1$. This is a constrained optimization problem. To solve it, we write down the corresponding Lagrangian:

$$L(\pi_i, \beta) = \sum_i \frac{\pi_i^2}{b_i} - 1 + \beta \left( \sum_i \pi_i - 1 \right)$$

We take the partial derivatives wrt $\pi_i$ and $\beta$ and set them to 0:

$$\frac{\partial L}{\partial \pi_i} = \pi_i \frac{2}{b_i} + \beta = 0 \Rightarrow \pi_i = -\frac{\beta b_i}{2} \tag{5}$$

$$\frac{\partial L}{\partial \beta} = \sum_i \pi_i - 1 = 0 \tag{6}$$

By taking (5) and plugging into (6), we get the following expression for $\beta$:

$$-\frac{\beta}{2} \sum_i b_i = 1 \Rightarrow \beta = -\frac{2}{\sum_i b_i}$$

By substituting $\beta$ into (5) we obtain:

$$\pi_i = \frac{b_i}{\sum_i b_i}$$

This is exactly the policy induced by the recognizer defined by $c(a_i) = 1$ iff $a_i \in A$. $\diamond$

We also note that it is advantageous, from the point of view of minimizing the variance of the updates, to have recognizers that accept a broad range of actions:

**Theorem 2** *Consider two binary recognizers $c_1$ and $c_2$, such that $\mu_1 > \mu_2$. Then the importance sampling corrections for $c_1$ have lower variance than the importance sampling corrections for $c_2$.*

**Proof:** From the previous theorem, we have the variance of a recognizer $c_A$:

$$Var = \sum_i \frac{\pi_i^2}{b_i} - 1 = \sum_i \left( \frac{b_i}{\sum_{j \in A} b_j} \right)^2 \frac{1}{b_i} - 1 = \frac{1}{\sum_{j \in A} b_j} - 1 = \frac{1}{\mu} - 1 \qquad \diamond$$

## 3 Formal framework for sequential problems

We turn now to the full case of learning about sequential decision processes with function approximation. We use the standard framework in which an agent interacts with a stochastic environment. At each time step $t$, the agent receives a state $s_t$ and chooses an action $a_t$. We assume for the moment that actions are selected according to a fixed behavior policy, $b : S \times \mathcal{A} \to [0,1]$ where $b(s,a)$ is the probability of selecting action $a$ in state $s$. The behavior policy is used to generate a sequence of experience (observations, actions and rewards). The goal is to learn, from this data, predictions about different ways of behaving. In this paper we focus on learning predictions about expected returns, but other predictions can be tackled as well (for instance, predictions of transition models for options (Sutton, Precup & Singh, 1999), or predictions specified by a TD-network (Sutton & Tanner, 2005; Sutton, Rafols & Koop, 2006)). We assume that the state space is large or continuous, and function approximation must be used to compute any values of interest. In particular, we assume a space of feature vectors $\Phi$ and a mapping $\phi : S \to \Phi$. We denote by $\phi_s$ the feature vector associated with $s$.

An option is defined as a triple $o = \langle I, \pi, \beta \rangle$ where $I \subseteq S$ is the set of states in which the option can be initiated, $\pi$ is the internal policy of the option and $\beta : S \to [0,1]$ is a stochastic termination condition. In the option work (Sutton, Precup & Singh, 1999), each of these elements has to be explicitly specified and fixed in order for an option to be well defined. Here, we will instead define options implicitly, using the notion of a recognizer.

A recognizer is defined as a function $c : S \times \mathcal{A} \to [0,1]$, where $c(s,a)$ indicates to what extent the recognizer allows action $a$ in state $s$. An important special case, which we treat in this paper, is that of binary recognizers. In this case, $c$ is an indicator function, specifying a subset of actions that are allowed, or recognized, given a particular state. Note that recognizers do not specify policies; instead, they merely give restrictions on the policies that are allowed or recognized.

A recognizer $c$ together with a behavior policy $b$ generates a *target policy* $\pi$, where:

$$\pi(s,a) = \frac{b(s,a)c(s,a)}{\sum_x b(s,x)c(s,x)} = \frac{b(s,a)c(s,a)}{\mu(s)} \tag{7}$$

The denominator of this fraction, $\mu(s) = \sum_x b(s,x)c(s,x)$, is the *recognition probability* at $s$, i.e., the probability that an action will be accepted at $s$ when behavior is generated according to $b$. The policy $\pi$ is only defined at states for which $\mu(s) > 0$. The numerator gives the probability that action $a$ is produced by the behavior and recognized in $s$. Note that if the recognizer accepts all state-action pairs, i.e. $c(s,a) = 1, \forall s, a$, then $\pi$ is the same as $b$.

Since a recognizer and a behavior policy can specify together a target policy, we can use recognizers as a way to specify policies for options, using (7). An option can only be initiated at a state for which at least one action is recognized, so $\mu(s) > 0, \forall s \in I$. Similarly, the termination condition of such an option, $\beta$, is defined as $\beta(s) = 1$ if $\mu(s) = 0$. In other words, the option must terminate if no actions are recognized at a given state. At all other states, $\beta$ can be defined between 0 and 1 as desired.

We will focus on computing the reward model of an option $o$, which represents the expected total return. The expected values of different features at the end of the option can be estimated similarly. The quantity that we want to compute is

$$E_o\{R(s)\} = E\{r_1 + r_2 + \ldots + r_T | s_0 = s, \pi, \beta\}$$

where $s \in I$, experience is generated according to the policy of the option, $\pi$, and $T$ denotes the random variable representing the time step at which the option terminates according to $\beta$. We assume that linear function approximation is used to represent these values, i.e.

$$E_o\{R(s)\} \approx \theta^T \phi_s$$

where $\theta$ is a vector of parameters.

## 4 Off-policy learning algorithm

In this section we present an adaptation of the off-policy learning algorithm of Precup, Sutton & Dasgupta (2001) to the case of learning about options. Suppose that an option's policy $\pi$ was used to generate behavior. In this case, learning the reward model of the option is a special case of temporal-difference learning of value functions. The forward view of this algorithm is as follows. Let $\bar{R}_t^{(n)}$ denote the truncated $n$-step return starting at time step $t$ and let $y_t$ denote the 0-step truncated return, $\bar{R}_t^{(0)}$. By the definition of the $n$-step truncated return, we have:

$$\bar{R}_t^{(n)} = r_{t+1} + (1 - \beta_{t+1})\bar{R}_{t+1}^{(n-1)}.$$

This is similar to the case of value functions, but it accounts for the possibility of terminating the option at time step $t+1$. The $\lambda$-return is defined in the usual way:

$$\bar{R}_t^{\lambda} = (1 - \lambda) \sum_{n=1}^{\infty} \lambda^{n-1} \bar{R}_t^{(n)}.$$

The parameters of the linear function approximator are updated on every time step proportionally to:

$$\Delta\bar{\theta}_t = \left[\bar{R}_t^{\lambda} - y_t\right] \nabla_\theta y_t (1 - \beta_1) \cdots (1 - \beta_t).$$

In our case, however, trajectories are generated according to the behavior policy $b$. The main idea of the algorithm is to use importance sampling corrections in order to account for the difference in the state distribution of the two policies.

Let $\rho_t = \frac{\pi(s_t, a_t)}{b(s_t, a_t)}$ be the importance sampling ratio at time step $t$. The truncated $n$-step return, $R_t^{(n)}$, satisfies:

$$R_t^{(n)} = \rho_t[r_{t+1} + (1 - \beta_{t+1})R_{t+1}^{(n-1)}].$$

The update to the parameter vector is proportional to:

$$\Delta\theta_t = \left[R_t^{\lambda} - y_t\right] \nabla_\theta y_t \rho_0 (1 - \beta_1) \cdots \rho_{t-1}(1 - \beta_t).$$

The following result shows that the expected updates of the on-policy and off-policy algorithms are the same.

**Theorem 3** *For every time step $t \geq 0$ and any initial state $s$,*

$$E_b[\Delta\theta_t|s] = E_\pi[\Delta\bar{\theta}_t|s].$$

**Proof:** First we will show by induction that $E_b\{R_t^{(n)}|s\} = E_\pi\{\bar{R}_t^{(n)}|s\}, \forall n$ (which implies that $E_b\{R_t^{\lambda}|s\} = E_\pi(\bar{R}_t^{\lambda}|s\})$. For $n = 0$, the statement is trivial. Assuming that it is true for $n-1$, we have

$$
\begin{aligned}
E_b\left\{R_t^{(n)}|s\right\} &= \sum_a b(s,a) \sum_{s'} P_{ss'}^a \rho(s,a) \left[r_{ss'}^a + (1 - \beta(s'))E_b\left\{R_{t+1}^{(n-1)}|s'\right\}\right] \\
&= \sum_a \sum_{s'} P_{ss'}^a b(s,a) \frac{\pi(s,a)}{b(s,a)} \left[r_{ss'}^a + (1 - \beta(s'))E_\pi\left\{\bar{R}_{t+1}^{(n-1)}|s'\right\}\right] \\
&= \sum_a \pi(s,a) \sum_{s'} P_{ss'}^a \left[r_{ss'}^a + (1 - \beta(s'))E_\pi\left\{\bar{R}_{t+1}^{(n-1)}|s'\right\}\right] = E_\pi\left\{\bar{R}_t^{(n)}|s\right\}.
\end{aligned}
$$

Now we are ready to prove the theorem's main statement. Defining $\Omega_t$ to be the set of all trajectory components up to state $s_t$, we have:

$$E_b\{\Delta\theta_t|s\} = \sum_{\omega \in \Omega_t} P_b(\omega|s) E_b\left\{(R_t^{\lambda} - y_t)\nabla_\theta y_t|\omega\right\} \prod_{i=0}^{t-1} \rho_i(1 - \beta_{i+1})$$

$$= \sum_{\omega \in \Omega_t} \left( \prod_{i=0}^{t-1} b_i P^{a_i}_{s_i s_{i+1}} \right) \left[ E_b \left\{ R_t^\lambda | s_t \right\} - y_t \right] \nabla_\theta y_t \prod_{i=0}^{t-1} \frac{\pi_i}{b_i} (1 - \beta_{i+1})$$

$$= \sum_{\omega \in \Omega_t} \left( \prod_{i=0}^{t-1} \pi_i P^{a_i}_{s_i s_{i+1}} \right) \left[ E_\pi \left\{ \bar{R}_t^\lambda | s_t \right\} - y_t \right] \nabla_\theta y_t (1 - \beta_1) ... (1 - \beta_t)$$

$$= \sum_{\omega \in \Omega_t} P_\pi(\omega|s) E_\pi \left\{ (\bar{R}_t^\lambda - y_t) \nabla_\theta y_t | \omega \right\} (1 - \beta_1) ... (1 - \beta_t) = E_\pi \left\{ \Delta \bar{\theta}_t | s \right\} .$$

Note that we are able to use $s_t$ and $\omega$ interchangeably because of the Markov property. $\diamond$

Since we have shown that $E_b[\Delta\theta_t|s] = E_\pi[\Delta\bar{\theta}_t|s]$ for any state $s$, it follows that the expected updates will also be equal for any distribution of the initial state $s$. When learning the model of options with data generated from the behavior policy $b$, the starting state distribution with respect to which the learning is performed, $I_0$ is determined by the stationary distribution of the behavior policy, as well as the initiation set of the option $I$. We note also that the importance sampling corrections only have to be performed for the trajectory since the initiation of the updates for the option. No corrections are required for the experience prior to this point. This should generate updates that have significantly lower variance than in the case of learning values of policies (Precup, Sutton & Dasgupta, 2001).

Because of the termination condition of the option, $\beta$, $\Delta\theta$ can quickly decay to zero. To avoid this problem, we can use a *restart function* $g : S \rightarrow [0, 1]$, such that $g(s_t)$ specifies the extent to which the updating episode is considered to start at time $t$. Adding restarts generates a new forward update:

$$\Delta\theta_t = (R_t^\lambda - y_t) \nabla_\theta y_t \sum_{i=0}^{t} g_i \rho_i ... \rho_{t-1} (1 - \beta_{i+1}) ... (1 - \beta_t), \qquad (8)$$

where $R_t^\lambda$ is the same as above. With an adaptation of the proof in Precup, Sutton & Dasgupta (2001), we can show that we get the same expected value of updates by applying this algorithm from the original starting distribution as we would by applying the algorithm without restarts from a starting distribution defined by $I_0$ and $g$. We can turn this forward algorithm into an incremental, backward view algorithm in the following way:

- Initialize $k_0 = g_0, e_0 = k_0 \nabla_\theta y_0$
- At every time step $t$:

$$\begin{aligned}
\delta_t &= \rho_t \left( r_{t+1} + (1 - \beta_{t+1}) y_{t+1} \right) - y_t \\
\theta_{t+1} &= \theta_t + \alpha \delta_t e_t \\
k_{t+1} &= \rho_t k_t (1 - \beta_{t+1}) + g_{t+1} \\
e_{t+1} &= \lambda \rho_t (1 - \beta_{t+1}) e_t + k_{t+1} \nabla_\theta y_{t+1}
\end{aligned}$$

Using a similar technique to that of Precup, Sutton & Dasgupta (2001) and Sutton & Barto (1998), we can prove that the forward and backward algorithm are equivalent (omitted due to lack of space). This algorithm is guaranteed to converge if the variance of the updates is finite (Precup, Sutton & Dasgupta, 2001). In the case of options, the termination condition $\beta$ can be used to ensure that this is the case.

## 5   Learning when the behavior policy is unknown

In this section, we consider the case in which the behavior policy is unknown. This case is generally problematic for importance sampling algorithms, but the use of recognizers will allow us to define importance sampling corrections, as well as a convergent algorithm. Recall that when using a recognizer, the target policy of the option is defined as:

$$\pi(s, a) = \frac{c(s, a) b(s, a)}{\mu(s)}$$

and the recognition probability becomes:
$$\rho(s,a) = \frac{\pi(s,a)}{b(s,a)} = \frac{c(s,a)}{\mu(s)}$$

Of course, $\mu(s)$ depends on $b$. If $b$ is unknown, instead of $\mu(s)$, we will use a maximum likelihood estimate $\hat{\mu} : S \to [0,1]$. The structure used to compute $\hat{\mu}$ will have to be compatible with the feature space used to represent the reward model. We will make this more precise below. Likewise, the recognizer $c(s,a)$ will have to be defined in terms of the features used to represent the model. We will then define the importance sampling corrections as:
$$\hat{\rho}(s,a) = \frac{c(s,a)}{\hat{\mu}(s)}$$

We consider the case in which the function approximator used to model the option is actually a state aggregator. In this case, we will define recognizers which behave consistently in each partition, i.e., $c(s,a) = c(p,a), \forall s \in p$. This means that an action is either recognized or not recognized in all states of the partition. The recognition probability $\hat{\mu}$ will have one entry for every partition $p$ of the state space. Its value will be:
$$\hat{\mu}(p) = \frac{N(p,c=1)}{N(p)}$$

where $N(p)$ is the number of times partition $p$ was visited, and $N(p,c=1)$ is the number of times the action taken in $p$ was recognized. In the limit, w.p.1, $\hat{\mu}$ converges to $\sum_s d^b(s|p) \sum_a c(p,a)b(s,a)$ where $d^b(s|p)$ is the probability of visiting state $s$ from partition $p$ under the stationary distribution of $b$. At this limit, $\hat{\pi}(s,a) = \hat{\rho}(s,a)b(s,a)$ will be a well-defined policy (i.e., $\sum_a \hat{\pi}(s,a) = 1$). Using Theorem 3, off-policy updates using importance sampling corrections $\hat{\rho}$ will have the same expected value as on-policy updates using $\hat{\pi}$. Note though that the learning algorithm never uses $\hat{\pi}$; the only quantities needed are $\hat{\rho}$, which are learned incrementally from data.

For the case of general linear function approximation, we conjecture that a similar idea can be used, where the recognition probability is learned using logistic regression. The development of this part is left for future work.

### Acknowledgements

The authors gratefully acknowledge the ideas and encouragement they have received in this work from Eddie Rafols, Mark Ring, Lihong Li and other members of the rlai.net group. We thank Csaba Szepesvari and the reviewers of the paper for constructive comments. This research was supported in part by iCore, NSERC, Alberta Ingenuity, and CFI.

### References

Baird, L. C. (1995). Residual algorithms: Reinforcement learning with function approximation. In *Proceedings of ICML*.

Precup, D., Sutton, R. S. and Dasgupta, S. (2001). Off-policy temporal-difference learning with function approximation. In *Proceedings of ICML*.

Sutton, R.S., Precup D. and Singh, S (1999). Between MDPs and semi-MDPs: A framework for temporal abstraction in reinforcement learning. *Artificial Intelligence*, vol . 112, pp. 181–211.

Sutton,, R.S. and Tanner, B. (2005). Temporal-difference networks. In *Proceedings of NIPS-17*.

Sutton R.S., Raffols E. and Koop, A. (2006). Temporal abstraction in temporal-difference networks". In *Proceedings of NIPS-18*.

Tadic, V. (2001). On the convergence of temporal-difference learning with linear function approximation. In *Machine learning* vol. 42, pp. 241-267.

Tsitsiklis, J. N., and Van Roy, B. (1997). An analysis of temporal-difference learning with function approximation. *IEEE Transactions on Automatic Control 42*:674–690.
